# BASINS OF ATTRACTION FOR ELECTRONIC NEURAL NETWORKS

C. M. Marcus
R. M. Westervelt
Division of Applied Sciences and Department of Physics
Harvard University, Cambridge, MA 02138

## ABSTRACT

We have studied the basins of attraction for fixed point and oscillatory attractors in an electronic analog neural network. Basin measurement circuitry periodically opens the network feedback loop, loads raster-scanned initial conditions and examines the resulting attractor. Plotting the basins for fixed points (memories), we show that overloading an associative memory network leads to irregular basin shapes. The network also includes analog time delay circuitry, and we have shown that delay in symmetric networks can introduce basins for oscillatory attractors. Conditions leading to oscillation are related to the presence of frustration; reducing frustration by diluting the connections can stabilize a delay network.

## (1) - INTRODUCTION

The dynamical system formed from an interconnected network of nonlinear neuron-like elements can perform useful parallel computation[1-5]. Recent progress in controlling the dynamics has focussed on algorithms for encoding the location of fixed points[1,4] and on the stability of the flow to fixed points[3,5-8]. An equally important aspect of the dynamics is the structure of the basins of attraction, which describe the location of all points in initial condition space which flow to a particular attractor[10,22].

In a useful associative memory, an initial state should lead reliably to the "closest" memory. This requirement suggests that a well-behaved basin of attraction should evenly surround its attractor and have a smooth and regular shape. One dimensional basin maps plotting "pull in" probability against Hamming distance from an attractor do not reveal the shape of the basin in the high dimensional space of initial states[9,19]. Recently, a numerical study of a Hopfield network with discrete time and two-state neurons showed rough and irregular basin shapes in a two dimensional Hamming space, suggesting that the high dimensional basin has a complicated structure[10]. It is not known how the basin shapes change with the size of the network and the connection rule.

We have investigated the basins of attraction in a network with continuous state dynamics by building an electronic neural network with eight variable gain sigmoid neurons and a three level (+,0,-) interconnection matrix. We have also built circuitry that can map out the basins of attraction in two dimensional slices of initial state space (Fig.1). The network and the basin measurements are described in section 2.

In section 3, we show that the network operates well as an associative memory and can retrieve up to four memories (eight fixed points) without developing spurious attractors, but that for storage of three or more memories, the basin shapes become irregular.

In section 4, we consider the effects of time delay. Real network components cannot switch infinitely fast or propagate signals instantaneously, so that delay is an intrinsic part of any hardware implementation of a neural network. We have included a controllable CCD (charge coupled device) analog time delay in each neuron to investigate how time delay affects the dynamics of a neural network. We find that networks with symmetric interconnection matrices, which are guaranteed to converge to fixed points for no delay, show collective sustained oscillations when time delay is present. By discovering which configurations are maximally unstable to oscillation, and looking at how these configurations appear in networks, we are able to show that by diluting the interconnection matrix, one can reduce or eliminate the oscillations in neural networks with time delay.

## (2) - NETWORK AND BASIN MEASUREMENT

A block diagram of the network and basin measurement circuit is shown in fig.1.

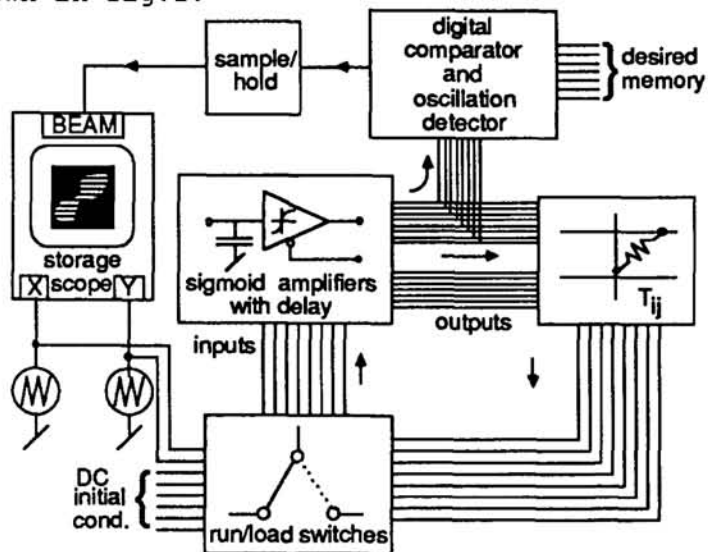

Fig.1 Block diagram of the network and basin measurement system.

The main feedback loop consists of non-linear amplifiers ("neurons", see fig.2) with capacitive inputs and a resistor matrix allowing interconnection strengths of $-1/R$, 0, $+1/R$ ($R = 100$ k$\Omega$). In all basin measurements, the input capacitance was 10 nF, giving a time constant of 1 ms. A charge coupled device (CCD) analog time delay[11] was built into each neuron, providing an adjustable delay per neuron over a range 0.4 - 8 ms.

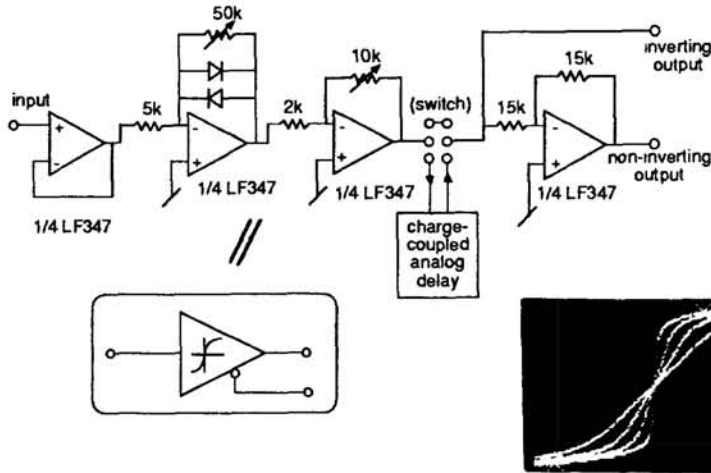

Fig.2 Electronic neuron. Non-linear gain provided by feedback diodes. Inset: Nonlinear behavior at several different values of gain.

Analog switches allow the feedback path to be periodically disconnected and each neuron input charged to an initial voltage. The network is then reconnected and settles to the attractor associated with that set of initial conditions. Two of the initial voltages are raster scanned (on a time scale that is long compared to the load/run switching time) with function generators that are also connected to the X and Y axes of a storage scope. The beam of the scope is activated when the network settles into a desired attractor, producing an image of the basin for that attractor in a two-dimensional slice of initial condition space. The "attractor of interest" can be one of the $2^8$ fixed points or an oscillatory attractor.

A simple example of this technique is the case of three neurons with symmetric non-inverting connection shown in fig.3.

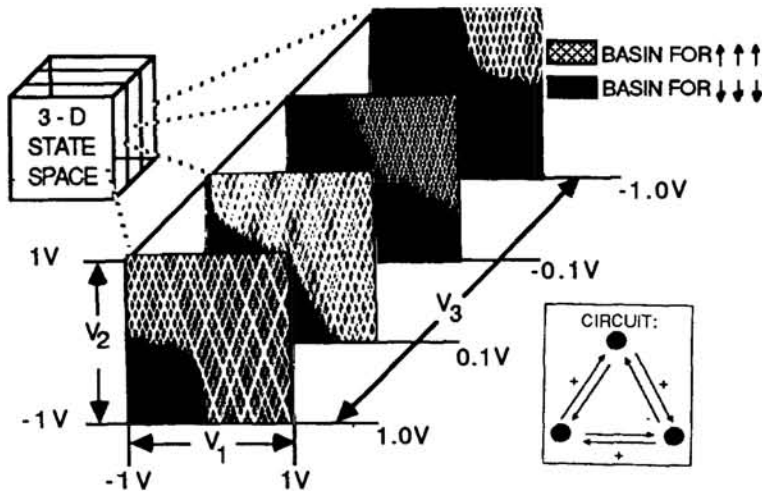

Fig.3 Basin of attraction for three neurons with symmetric non-inverting coupling. Slices are in the plane of initial voltages on neurons 1 and 2. The two fixed points are all neurons saturated positive or all negative. The data are photographs of the scope screen.

(3) BASINS FOR FIXED POINTS - ASSOCIATIVE MEMORY

Two dimensional slices of the eight dimensional initial condition space (for the full network) reveal important qualitative features about the high dimensional basins. Fig. 4 shows a typical slice for a network programmed with three memories according to a clipped Hebb rule[1,12]:

$$T_{ij} = 1/R \ \mathrm{Sgn}(\textstyle\sum_{\alpha=1,m} \xi_i{}^\alpha \xi_j{}^\alpha); \quad T_{ii} = 0 \qquad (1)$$

where $\xi$ is an N-component memory vector of 1's and -1's, and m is the number of memories. The memories were chosen to be orthogonal $(\xi^\alpha \cdot \xi^\beta = N \ \delta_{\alpha\beta})$.

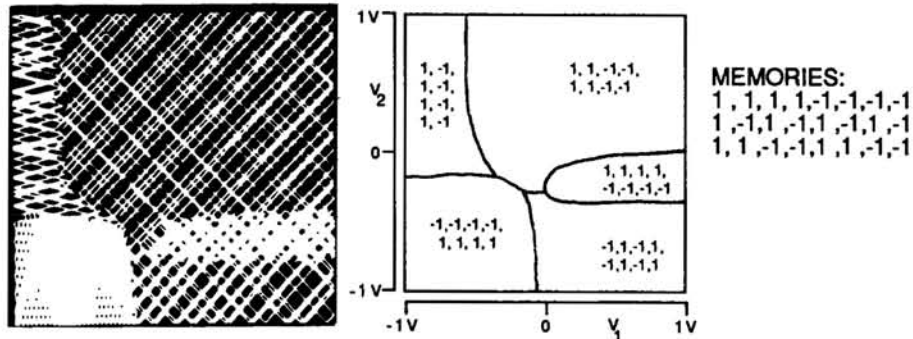

Fig. 4 A slice of initial condition space shows the basins of attraction for five of the six fixed points for three memories in eight-neuron Hopfield net. Learning rule was clipped Hebb (Eq.1). Neuron gain = 15.

Because the Hebb rule (eq.1) makes $\xi^\alpha$ and $-\xi^\alpha$ stable attractors, a three-memory network will have six fixed point attractors. In fig.4, the basins for five of these attractors are visible, each produced with a different rastering pattern to make it distinctive. Several characteristic features should be noted:

-- All initial conditions lead to one of the memories (or inverses), no spurious attractors were seen for three or four memories. This is interesting in light of the well documented emergence of spurious attractors at m/N ~15% in larger networks with discrete time[2,18].

-- The basins have smooth and continuous edges.

-- The shapes of the basins as seen in this slice are irregular. Ideally, a slice with attractors at each of the corners should have rectangular basins, one basin in each quadrant of the slice and the location of the lines dividing quadrants determined by the initial conditions on the other neurons (the "unseen" dimensions). With three or more memories the actual basins do not resemble this ideal form.

(4) TIME DELAY, FRUSTRATION AND SUSTAINED OSCILLATION

Arguments defining conditions which guarantee convergence to fixed points[3,5,6] (based, for example, on the construction of a Liapunov function) generally assume instantaneous communication between elements of the network. In any hardware implementation, these assumptions break down due to the finite switching speed of amplifiers and the charging time of long interconnect lines.[13] It is the ratio of delay/RC which is important for stability, so keeping this ratio small limits how fast a neural network chip can be designed to run. Time delay is also relevant to biological neural nets where propagation and response times are comparable.[14,15]

Our particular interest in this section is how time delay can lead to sustained oscillation in networks which are known to be stable when there is no delay. We therefore restrict our attention to networks with symmetric interconnection matrices ($T_{ij} = T_{ji}$).

An obvious ingredient in producing oscillations in a delay network is feedback, or stated another way, a graph representing the connections in a network must contain loops.

The simplest oscillatory structure made of delay elements is the ring oscillator (fig.5a). Though not a symmetric configuration, the ring oscillator illustrates an important point: the ring will oscillate only when there is negative feedback at dc - that is, when the product of interconnection around the loop is negative. Positive feedback at dc (loop product of connections > 0) will lead to saturation.

Observing various symmetric configurations (e.g. fig.5b) in the delayed-neuron network, we find that a negative product of connections around a loop is also a necessary condition for sustained oscillation in symmetric circuits. An important difference between the ring (fig.5a) and the symmetric loop (fig.5b) is that the period of oscillation for the ring is the total accumulated delay around the ring - the larger the ring the longer the period. In contrast, for those symmetric configurations which have oscillatory attractors, the period of oscillation is roughly twice the delay, regardless of the size of the configuration or the value of delay. This indicates that for symmetric configurations the important feedback path is local, not around the loop.

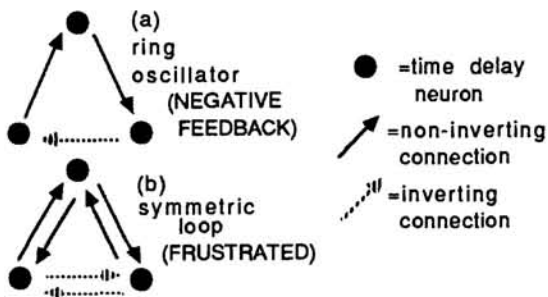

(a) ring oscillator (NEGATIVE FEEDBACK)

● =time delay neuron

→ =non-inverting connection

⤏ =inverting connection

(b) symmetric loop (FRUSTRATED)

Fig.5 (a) A ring oscillator: needs negative feedback at dc to oscillate. (b) Symmetrically connected triangle. This configuration is "frustrated" (defined in text), and has both oscillatory and fixed point attractors when neurons have delay.

Configurations with loop connection product < 0 are important in the theory of spin glasses[16], where such configurations are called "frustrated." Frustration in magnetic (spin) systems, gives a measure of "serious" bond disorder (disorder that cannot be removed by a change of variables) which can lead to a spin glass state.[16,17] Recent results based on the similarity between spin glasses and symmetric neural networks has shown that storage capacity limitations can be understood in terms of this bond disorder.[18,19] Restating our observation above: We only find stable oscillatory modes in symmetric networks with delay when there is frustration. A similar result for a sign-symmetric network ($T_{ij}$, $T_{ji}$ both $\geq 0$ or $\leq 0$) with no delay is described by Hirsch.[6]

We can set up the basin measurement system (fig.1) to plot the basin of attraction for the oscillatory mode. Fig.6 shows a slice of the oscillatory basin for a frustrated triangle of delay neurons.

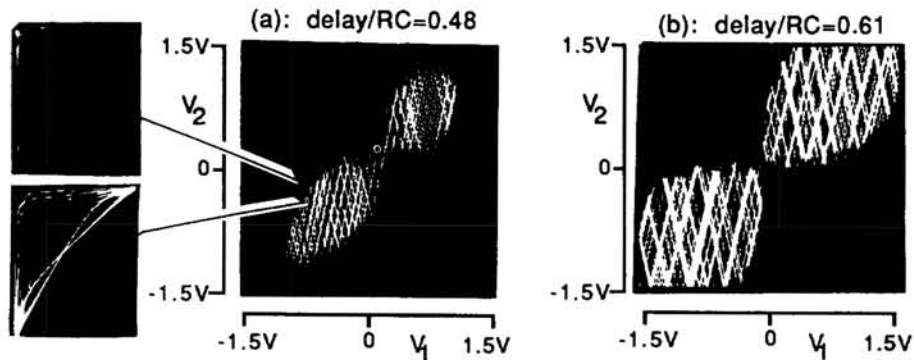

Fig.6 Basin for oscillatory attractor (cross-hatched region) in frustrated triangle of delay-neurons. Connections were all symmetric and inverting; other frustrated configurations (e.g. two non-inverting, one inverting, all symmetric) were similar. (6a): delay = 0.48RC, inset shows trajectory to fixed point and oscillatory mode for two close-lying initial conditions. (6b): delay = 0.61RC, basin size increases.

A fully connected feedback associative network with more that one memory will contain frustration. As more memories are added, the amount of frustration will increases until memory retrieval disappears. But before this point of memory saturation is reached, delay could cause an oscillatory basin to open. In order to design out this possibility, one must understand how frustration, delay and global stability are related. A first step in determining the stability of a delay network is to consider which small configurations are most prone to oscillation, and then see how these "dangerous" configurations show up in the network. As described above, we only need to consider frustrated configurations.

A frustrated configuration of neurons can be sparsely connected, as in a loop, or densely connected, with all neurons connected to all others, forming what is called in graph theory a "clique." Representing a network with inverting and non-inverting connections as a signed graph (edges carry + and -), we define a *frustrated clique* as a fully connected set of vertices (r vertices; $r(r-1)/2$ edges) with all sets of three vertices in the clique forming frustrated triangles. Some examples of frustrated loops and cliques are shown in fig.7. Notice that neurons connected with all inverting symmetric connections, a configuration that is useful as a "winner-take-all" circuit, is a frustrated clique.

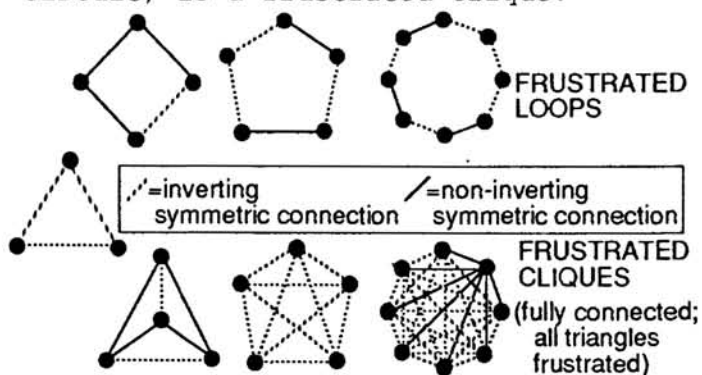

Fig.7 Examples of frustrated loops and frustrated cliques. In the graph representation vertices (black dots) are neurons (with delay) and undirected edges are symmetric connections.

We find that delayed neurons connected in a frustrated loop
longer than three neurons do not show sustained oscillation for any
value of delay (tested up to delay = 8RC). In contrast, when delayed
neurons are connected in any frustrated clique configuration, we do
find basins of attraction for sustained oscillation as well as fixed
point attractors, and that the larger the frustrated clique, the more
easily it oscillates in the following ways: (1) For a given value of
delay/RC, the size of the oscillatory basin increases with r, the
size of the frustrated clique (fig.8). (2) The critical value of
delay at which the volume of the oscillatory basin goes to zero
decreases with increasing r (fig.9); For r=8 the critical delay is
already less than 1/30 RC.

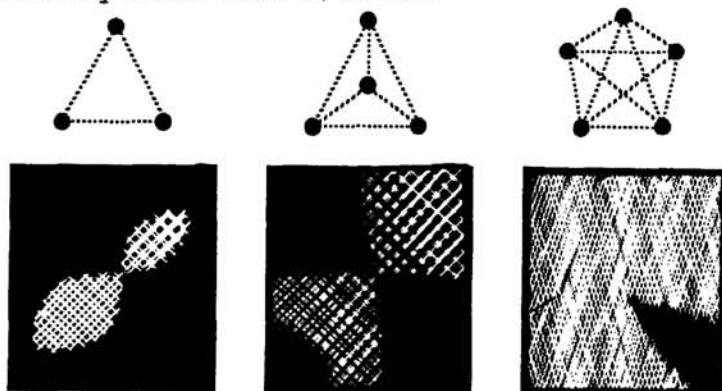

Fig.8 Size of basin
for oscillatory mode
increases with size of
frustrated clique. The
delay is 0.46RC per
neuron in each picture.
Slices are in the space
of initial voltages on
neurons 1 and 2, other
initial voltages near
zero.

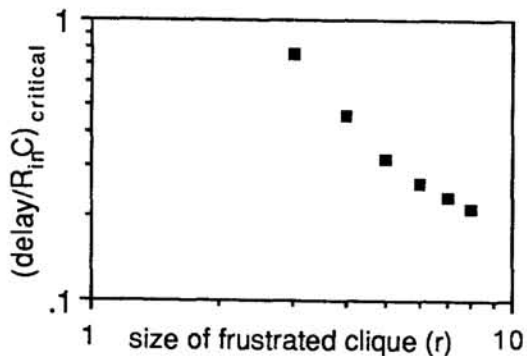

Fig.9 The critical value of delay
where the oscillatory mode vanishes.
Measured by reducing delay until
system leaves oscillatory attractor.
Delay plotted in units of the
characteristic time $R_{in}C$, where $R_{in}$
$=(\Sigma_j\ 1/R_{ij})^{-1}=10^5\Omega/(r-1)$ and C=10nF,
indicating that the critical delay
decreases faster than $1/(r-1)$.

Having identified frustrated cliques as the maximally unstable
configuration of time delay neurons, we now ask how many cliques of a
given size do we expect to find in a large network.

A set of r vertices (neurons) can be fully connected by $r(r-1)/2$
edges of two types (+ or -) to form $2^{r(r-1)/2}$ different cliques. Of
these, $2^{(r-1)}$ will be frustrated cliques. Fig.10 shows all $2^{(4-1)}=8$
cases for r=4.

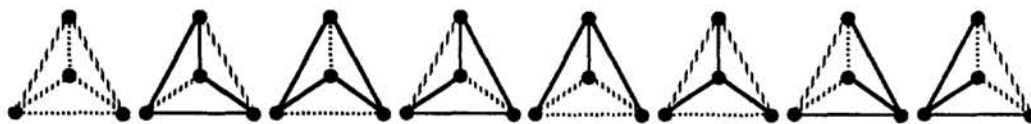

Fig.10 All graphs of size r=4 that are frustrated cliques
(fully connected, every triangle frustrated.) Solid lines =
positive edges, dashed lines = negative edges.

For a randomly connected network, this result combined with results from random graph theory[20] gives an expected number of frustrated cliques of size r in a network of size N, $E_N(r)$:

$$E_N(r) = \binom{N}{r} c(r,p) \qquad (2)$$

$$c(r,p) = 2^{-(r-1)(r-2)/2} p^{r(r-1)/2} \qquad (3)$$

where $\binom{N}{r}$ is the binomial coefficient and $c(r,p)$ is defined as the concentration of frustrated cliques. p is the connectance of the network, defined as the probability that any two neurons are connected. Eq.3 is the special case where + and – edges (non-inverting, inverting connections) are equally probable. We have also generalized this result to the case $p(+) \neq p(-)$.

Fig.11 shows the dramatic reduction in the concentration of all frustrated configurations in a diluted random network. For the general case $(p(+) \neq p(-))$ we find that the negative connections affect the concentrations of frustrated cliques more strongly than the positive connections, as expected (Frustration requires negatives, not positives, see fig.10).

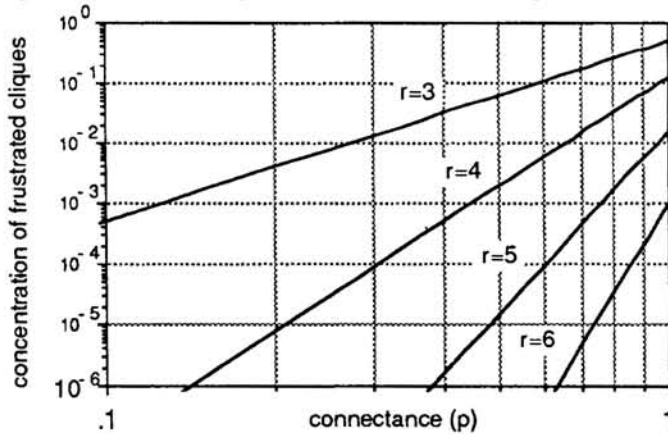

Fig.11 Concentration of frustrated cliques of size r=3,4,5,6 in an unbiased random network, from eq.3. Concentrations decrease rapidly as the network is diluted, especially for large cliques (note: log scale).

When the interconnections in a network are specified by a learning rule rather than at random, the expected numbers of any configuration will differ from the above results. We have compared the number of frustrated triangles in large three-valued (+1,0,-1) Hebb interconnection matrices (N=100,300,600) to the expected number in a random matrix of the same size and connectance. The Hebb matrix was constructed according to the rule:

$$T_{ij} = Z_k \left( \sum_{\alpha=1,m} \xi_i^\alpha \xi_j^\alpha \right) \; ; \; T_{ii} = 0 \qquad (4a)$$

$$Z_k(x) = +1 \text{ for } x > k; \; 0 \text{ for } -k \leq x \leq k; \; -1 \text{ for } x < -k; \qquad (4b)$$

m is the number of memories, $Z_k$ is a threshold function with cutoff k, and $\xi^\alpha$ is a random string of 1's and -1's. The matrix constructed by eq.4 is roughly unbiased (equal number of positive and negative connections) and has a connectance p(k). Fig.12 shows the ratio of frustrated triangles in a diluted Hebb matrix to the expected number in a random graph with the same connectance for different numbers of

memories stored in the Hebb matrix. At all values of connectance, the Hebb matrix has fewer frustrated triangles than the random matrix by a ratio that is decreased by diluting the matrix or storing fewer memories. The curves do not seem to depend on the size of the matrix, N. This result suggests that diluting a Hebb matrix breaks up frustration even more efficiently than diluting a random matrix.

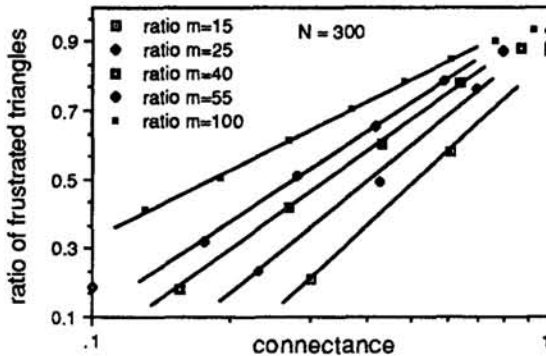

Fig.12 The number of frustrated triangles in a (+,0,-) Hebb rule matrix (300x300) divided by the expected number in a random signed graph with equal connectance. The different sets of points are for different numbers of random memories in the Hebb matrix. The lines are guides to the eye.

The sensitive dependence of frustration on connectance suggests that oscillatory modes in a large neural network with delay can be eliminated by diluting the interconnection matrix. As an example, consider a unbiased random network with delay = RC/10. From fig.9, only frustrated cliques of size r=5 or larger have oscillatory basins for this value of delay; frustration in smaller configurations in the network cannot lead to sustained oscillation in the network. Diluting the connectance to 60% will reduce the concentration of frustrated cliques with r=5 by a factor of over 100 and r=6 by a factor of 2000. The reduction would be even greater for a clipped Hebb matrix.

Results from spin glass theory[21] suggest that diluting a clipped Hebb matrix can actually improve the storage capacity for moderated dilution, with a maximum in the capacity at a connectance of 61%. To the extent this treatment applies to an analog continuous-time network, we should expect that by diluting connections, oscillatory modes can be killed before memory capacity is compromised.

We have confirmed the stabilizing effect of dilution in our network: For a fully connected eight neuron network programmed with three orthogonal memories according to eq.1, adding a delay of 0.4RC opens large basins for sustained oscillation. By randomly diluting the interconnections to p~0.85, we were able to close the oscillatory basins and recover a useful associative memory.

<center>SUMMARY</center>

We have investigated the structure of fixed point and oscillatory basins of attraction in an electronic network of eight non-linear amplifiers with controllable time delay and a three value (+,0,-) interconnection matrix.

For fixed point attractors, we find that the network performs well as an associative memory - no spurious attractors were seen for up to four stored memories - but for three or more memories, the shapes of the basins of attraction became irregular.

A network which is stable with no delay can have basins for oscillatory attractors when time delay is present. For symmetric networks with time delay, we only observe sustained oscillation when there is frustration. Frustrated cliques (fully connected configurations with all triangles frustrated), and not loops, are most prone to oscillation, and the larger the frustrated clique, the more easily it oscillates. The number of these "dangerous" configurations in a large network can be greatly reduced by diluting the connections. We have demonstrated that a network with a large basin for an oscillatory attractor can be stabilized by dilution.

## ACKNOWLEDGEMENTS

We thank K.L.Babcock, S.W.Teitsworth, S.Strogatz and P.Horowitz for useful discussions. One of us (C.M.M) acknowledges support as an AT&T Bell Laboratories Scholar. This work was supported by JSEP contract no. N00014-84-K-0465.

## REFERENCES

1) J.S.Denker, Physica 22D, 216 (1986).
2) J.J. Hopfield, Proc.Nat.Acad.Sci. 79, 2554 (1982).
3) J.J. Hopfield, Proc.Nat.Acad.Sci. 81, 3008 (1984).
4) J.S. Denker, Ed. Neural Networks for Computing, AIP Conf. Proc. 151 (1986).
5) M.A. Cohen, S. Grossberg, IEEE Trans. SMC-13, 815 (1983).
6) M.W.Hirsch, Convergence in Neural Nets, IEEE Conf.on Neural Networks, 1987.
7) K.L. Babcock, R.M. Westervelt, Physica 23D, 464 (1986).
8) K.L. Babcock, R.M. Westervelt, Physica 28D, 305 (1987).
9) See, for example: D.B.Schwartz, et al, Appl.Phys.Lett., 50 (16), 1110 (1987); or M.A.Silviotti,et al, in Ref.4, pg.408.
10) J.D. Keeler in Ref.4, pg.259.
11) CCD analog delay: EG&G Reticon RD5106A.
12) D.O.Hebb, The Organization of Behavior, (J.Wiley, N.Y., 1949).
13) Delay in VLSI discussed in: A. Muhkerjee, Introduction to nMOS and CMOS VLSI System Design, (Prentice Hall, N.J.,1985).
14) U. an der Heiden, J.Math.Biology, 8, 345 (1979).
15) M.C. Mackey, U. an der Heiden, J.Math.Biology,19, 221 (1984).
16) Theory of spin glasses reviewed in: K. Binder, A.P. Young, Rev. Mod. Phys.,58 (4),801, (1986).
17) E. Fradkin,B.A. Huberman,S.H. Shenker, Phys.Rev.B18 (9),4789 (1978).
18) D.J. Amit, H. Gutfreund, H. Sompolinski, Ann.Phys. 173, 30, (1987) and references therein.
19) J.L. van Hemmen, I. Morgenstern, Editors, Heidelberg Colloquium on Glassy Dynamics, Lecture Notes in Physics 275, (Springer-Verlag, Heidelberg, 1987).
20) P.Erdos, A.Renyi, Pub.Math.Inst.Hung.Acad.Sci., 5,17, (1960).
21) I.Morgenstern in Ref.19, pg.399;H.Sompolinski in Ref.19, pg.485.
22) J. Guckenheimer, P.Holmes, Nonlinear Oscillations,Dynamical Systems and Bifurcations of Vector Fields (Springer,N.Y.1983).
